# Predicting Weather Using a Genetic Memory: a Combination of Kanerva's Sparse Distributed Memory with Holland's Genetic Algorithms

David Rogers
Research Institute for Advanced Computer Science
MS 230-5, NASA Ames Research Center
Moffett Field, CA  94035

## ABSTRACT

Kanerva's sparse distributed memory (SDM) is an associative-memory model based on the mathematical properties of high-dimensional binary address spaces. Holland's genetic algorithms are a search technique for high-dimensional spaces inspired by evolutionary processes of DNA. "Genetic Memory" is a hybrid of the above two systems, in which the memory uses a genetic algorithm to dynamically reconfigure its physical storage locations to reflect correlations between the stored addresses and data. For example, when presented with raw weather station data, the Genetic Memory discovers specific features in the weather data which correlate well with upcoming rain, and reconfigures the memory to utilize this information effectively. This architecture is designed to maximize the ability of the system to scale-up to handle real-world problems.

## INTRODUCTION

The future success of neural networks depends on an ability to "scale-up" from small networks and low-dimensional toy problems to networks of thousands or millions of nodes and high-dimensional real-world problems. (The *dimensionality* of a problem refers to the number of variables needed to describe the problem domain.) Unless neural networks are shown to be scalable to real-world problems, they will likely remain restricted to a few specialized applications.

Scaling-up adds two types of computational demands to a system. First, there is a linear increase in computational demand proportional to the increased number of variables. Second, there is a greater, nonlinear increase in computational demand due to

the number of interactions that can occur between the variables. This latter effect is primarily responsible for the difficulties encountered in scaling-up many systems. In general, it is difficult to scale-up a system unless it is specifically designed to function well in high-dimensional domains.

Two systems designed to function well in high-dimensional domains are Kanerva's sparse distributed memory (Kanerva, 1988) and Holland's genetic algorithms (Holland, 1986). I hypothesized that a hybrid of these two systems would preserve this ability to operate well in high-dimensional environments, and offer grater functionality than either individually. I call this hybrid *Genetic Memory*. To test its capabilities, I applied it to the problem of forecasting rain from local weather data.

Kanerva's sparse distributed memory (SDM) is an associative-memory model based on the mathematical properties of high-dimensional binary address spaces. It can be represented as a three-layer neural-network with an extremely large number of nodes (1,000,000+) in the middle layer. In its standard formulation, the connections between the input layer and the hidden layer (the *input representation* used by the system) are fixed, and learning is done by changing the values of the connections between the hidden layer and the output layer.

Holland's genetic algorithms are a search technique for high-dimensional spaces inspired by evolutionary processes of DNA. Members of a set of binary strings competes for the opportunity to recombine. Recombination is done by selecting two "successful" members of the population to be the parents. A new string is created by splicing together pieces of each parent. Finally, the new string is placed into the set, and some "unsuccessful" older string removed.

"Genetic Memory" is a hybrid of the above two systems. In this hybrid, a genetic algorithm is used to reconfigure the connections between the input layer and the hidden layer. The connections between the hidden layer and the output layer are changed using the standard method for a sparse distributed memory. The "success" of an input representation is determined by how well it reflects correlations between addresses and data, using my previously presented work on statistical prediction (Rogers, 1988). Thus, we have two separate learning algorithms in the two levels. The memory uses the genetic algorithm to dynamically reconfigure its input representation to better reflect correlations between collections of input variables and the stored data.

I applied this Genetic Memory architecture to the problem of predicting rain given only local weather features such as the air pressure, the cloud cover, the month, the temperature, etc. The weather data contained 15 features, sampled every 4-hours over a 20-year period on the Australian coast. I coded each state into a 256-bit address, and stored at that address a single bit which denoted whether it rained in the 4 hours following that weather state. I allowed the genetic algorithm to reconfigure the memory while it scanned the file of weather states.

The success of this procedure was measured in two ways. First, once the training was completed, the Genetic Memory was better at predicting rain than was the standard sparse distributed memory. Second, I had access to the input representations discovered by the Genetic Memory and could view the specific combinations of features that predicted rain. Thus, unlike many neural networks, the Genetic Memory allows the user to inspect the internal representations it discovers during training.

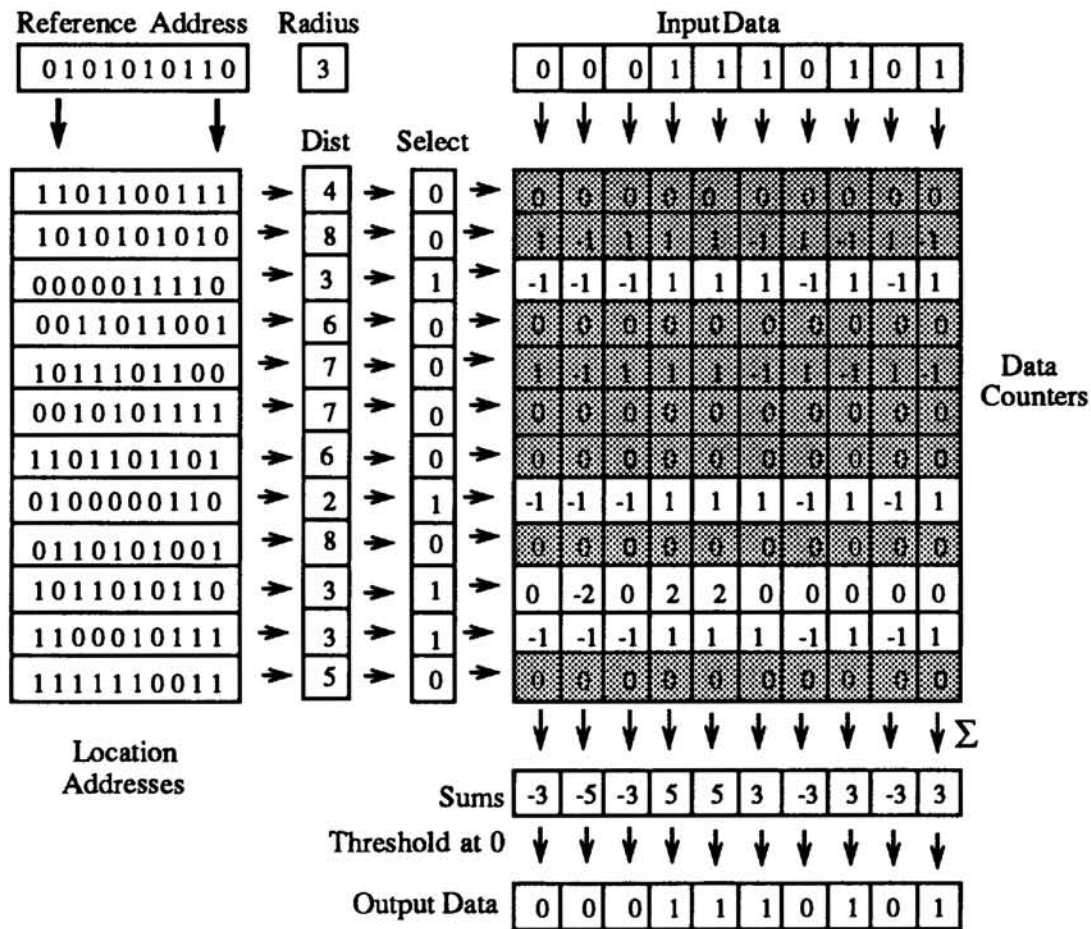

**Figure 1:** Structure of a Sparse Distributed Memory

## SPARSE DISTRIBUTED MEMORY

Sparse distributed memory can be best illustrated as a variant of random-access memory (RAM). The structure of a twelve-location SDM with ten-bit addresses and ten-bit data is shown in figure 1.

A *memory location* is a row in this figure. The *location addresses* are set to random addresses. The *data counters* are initialized to zero. All operations begin with *addressing* the memory; this entails finding the Hamming distance between the *reference address* and each of the location addresses. If this distance is less than or equal to the *Hamming radius*, the select-vector entry is set to 1, and that location is termed *selected*. The ensemble of such selected locations is called the *selected set*. Selection is noted in the figure as non-gray rows. A *radius* is chosen so that only a small percentage of the memory locations are selected for a given reference address.

When *writing* to the memory, all selected counters beneath elements of the *input data* equal to 1 are incremented, and all selected counters beneath elements of the *input data* equal to 0 are decremented. This completes a write operation. When *reading* from the memory, the selected data counters are summed columnwise into the register *sums*. If the value of a sum is greater than or equal to zero, we set the corresponding bit in the *output data* to 1; otherwise, we set the bit in the *output data* to 0. (When reading, the contents of the *input data* are ignored.)

This example makes clear that a datum is *distributed* over the data counters of the selected locations when writing, and that the datum is reconstructed during reading by *averaging* the sums of these counters. However, depending on what additional data were written into some of the selected locations, and depending on how these data correlate with the original data, the reconstruction may contain noise.

The SDM model can also be described as a fully-connected three-layer feed-forward neural network. In this model, the location addresses are the weights between the input layer and the hidden units, and the data counters are the weights between the hidden units and the output layer. Note that the number of hidden-layer nodes (at least 1,000 and possibly up to 1,000,000) is much larger than is commonly used for artificial neural networks. It is unclear how well standard algorithms, such as back-propagation, would perform with such a large number of units in the hidden layer.

## HOLLAND'S GENETIC ALGORITHMS

Genetic Algorithms are a search technique for high-dimensional spaces inspired by the evolutionary processes of DNA. The domain of a genetic algorithm is a *population of fixed-length binary strings* and a *fitness function*, which is a method for evaluating the fitness of each of the members. We use this fitness function to select two highly-ranked members for recombination, and one lowly-ranked member for replacement. (The selection may be done either *absolutely*, with the best and worst members always being selected, or *probabilisticly*, with the members being chosen proportional to their fitness scores.)

The member selected as *bad* is removed from the population. The two members selected as *good* are then recombined to create a new member to take its place in the population. In effect, the genetic algorithm is a search over a high-dimensional space for strings which are highly-rated by the fitness function.

The process used to create new members of the population is called *crossover*. In a crossover, we align the two good candidates end-to-end and segment them at one or more crossover-points. We then create a new string by starting the transcription of bits at one of the parent strings, and switching the transcription to the other parent at the crossover-points. This new string is placed into the population, taking the place of the poorly-rated member.

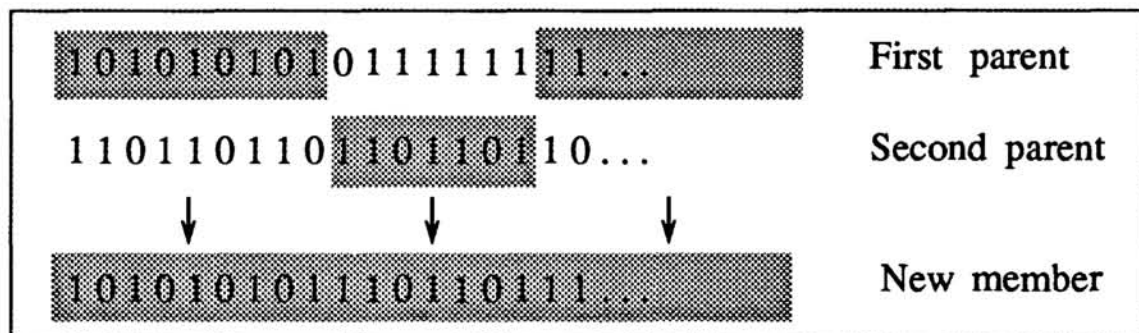

Figure 2: Crossover of two binary strings

By running the genetic algorithm over the population many times, the population "evolves" towards members which are rated more fit by our fitness function.

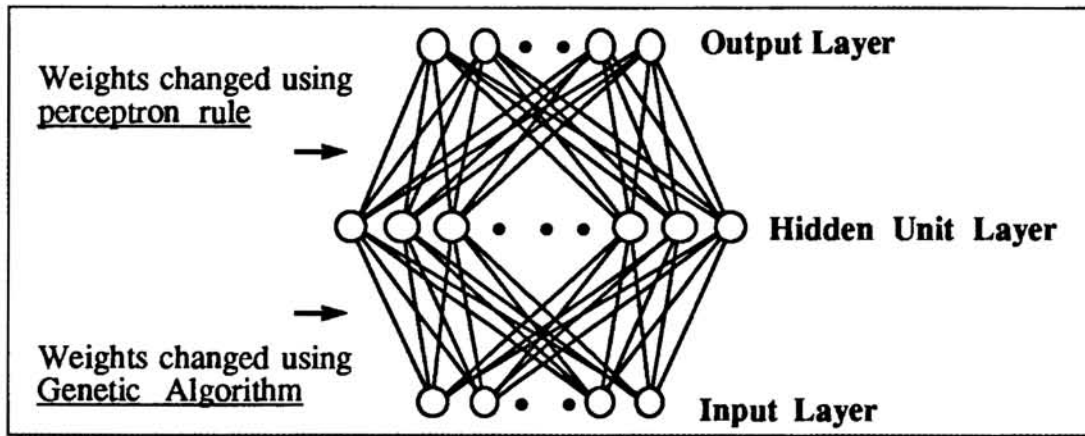

**Figure 3**: Structure of a Genetic Memory

Holland has a mathematical proof that genetic algorithms based on the crossover procedure are an extremely efficient method for searching a high-dimensional space.

## GENETIC MEMORY

Genetic Memory is a hybrid of Kanerva's Sparse Distributed Memory and Holland's Genetic Algorithms. In this hybrid, the location addresses of the SDM are not held constant; rather, a Genetic Algorithm is used to move them to more advantageous positions in the address space. If we view SDM as a neural net, this hybrid uses a genetic algorithm to change the weights in the connections between the input layer and the hidden unit layer, while the connections between the hidden unit layer and the output layer at changed using the standard method for a SDM.

Most other work which combined neural networks and genetic algorithms kept *multiple* networks; the Genetic Algorithm was used to recombine the more successful of these networks to create new entire networks.

In a Genetic Memory there is a *single* network with different algorithms changing the weights in different layers. Thus, a Genetic Memory incorporates the Genetic Algorithm directly into the operation of a single network.

## AUSTRALIAN WEATHER DATA

The weather data was collected at a single site on the Australian coast. A sample was taken every 4 hours for 25 years; the file contains over 58,000 weather samples

The file contained 15 distinct features, including year, month, day of the month, time of day, pressure, dry bulb temperature, wet bulb temperature, dew point, wind speed, wind direction, cloud cover, and whether it rained in the past four hours.

For this work, I coded each weather sample into a 256-bit word. Each weather sample was coded into a 256-bit binary address, giving each feature a 16-bit field in that address. The feature values were coarse-coded into a simple thermometer-style code. For example, figure 4 shows the code used for month.

## PROCEDURE FOR WEATHER PREDICTION

In the standard SDM model, the locations addresses are held constant. In a Genetic Memory, the location addresses are reconfigured using a Genetic Algorithm.

```
JAN: 1111111100000000       JUL: 1000000001111111
FEB: 0111111111000000       AUG: 1100000000111111
MAR: 0011111111100000       SEP: 1111000000011111
APR: 0000111111110000       OCT: 1111100000001111
MAY: 0000011111111000       NOV: 1111110000000011
JUN: 0000001111111110       DEC: 1111111000000001
```

**Figure 4**: 16-bit code used for month

The fitness function used is based on my work on statistical prediction and presented at NIPS-88 [Rogers 1988]. This work assigns a number to each physical storage location (a row in the figure) which is a measure of the *predictiveness* of that location. Highly-predictive locations are recombined using crossover; the newly-created location address is given to a location which is relatively unpredictive. *The data counter is a measure of the correlation between the selection of a location and the occurrence of a given bit value.* Thus, we can use the data counters to judge the fitness, i.e., the predictiveness, of each memory location.

To train the memory, we present the memory with each weather state in turn. The memory is *not* shown the data a multiple number of times. For each state, the memory is addressed with the 256-bit address which represents it. **"0"** is written to the memory if it does not rain in the next four hours, and **"1"** if it does. After the memory has seen a given number of weather samples, the Genetic Algorithm is performed to replace a poorly-predictive location with a new address created from two predictive addresses.

The procedure is continued until the memory has seen 50,000 weather samples, and has performed ~5,000 genetic recombinations.

## ANALYSIS OF RESULTS

The initial results from the Genetic Memory procedure was conducted on a memory with 1,000 storage locations. The weather sample set consisted of a sequence of weather samples taken every 4 hours over a period of 20 years. In the sample set, it rained in the next 4 hours for ~10% of the samples, and was dry in the next four hours in ~90% of the samples.

The Genetic Memory was testing by storing ~50,000 weather samples. The samples were given to the memory in chronological order. During the course of storage, the memory reconfigured itself with ~5,000 genetic recombinations. A Genetic Memory and a standard Sparse Distributed Memory were tested against 1,000 previously unseen weather samples. In initial experiments, the Genetic Memory had 50% fewer errors than the Sparse Distributed Memory.

However, the Genetic Memory does not only show an improvement in performance, it allows the user to analyze the genetically-determined memory locations to discover how the memory improved its performance.

By studying highly-rated memory locations in the Genetic Memory, we can *open the black box:* that is, access the parameters the memory has decided are the most effective in associating the sample addresses with the sample data. This ability to access the parameters the system found effective has two important implications. First,

the parameters may offer insights into the underlying physical processes in the system under study. Second, knowledge of *how* the system predicts may be vital for determining the robustness and the envelope of applicability of the memory prior to embedding into a real-world system.

Simply scoring the performance of a system is not enough. We must be able to "open the black box" to study why the system performs as it does.

## OPENING THE BLACK BOX

When the training is completed, we can analyze the structure of memory locations which performed well to discover *which features* they found most discriminatory and *which values* of those features were preferred. For example, here is a memory location which was rated highly-fit for predicting rain after training:

1101001100000011 1111011110101011 0111111100010000 1100000011011010
0100110011111011 1111110000000011 0111111011000000 0011101101100110
0000001011110110 0110000001000010 0001001110110100 0100001111111111
0000000111111110 0000000011111111 0011011111111111 0100110000001000

By measuring the distance between a given 16-bit field and all possible values for that field, we can discover *which values* of the feature are most desired. (Closer in hamming distance is better.) The absolute range of values is the *sensitivity* of the location to changes along that feature dimension. Figure 5 shows an analysis of the 16-bit field for month in the given memory location:

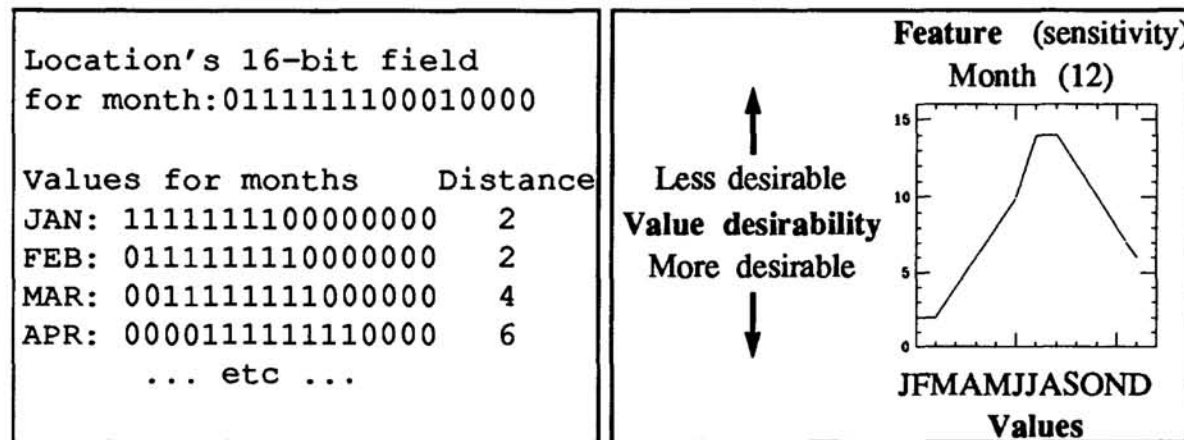

**Figure 5**: Analyzing a location field

In this case, the location finds January and February the most desirable months for rain, and July and August the least desirable months.

The relative sensitivity towards different features measures which features are most important in making the prediction of rain. In this case, we have a change of distance of 12 bits, which makes this location very sensitive to the value of the month.

We can estimate which features are the most important in predicting rain by looking at the relative sensitivity of the different fields in the location to changes in their feature. The following graphs show the most sensitive features of the previously shown memory location towards predicting rain; that is, the location is very sensitive to the combination of all these fields with the proper values.

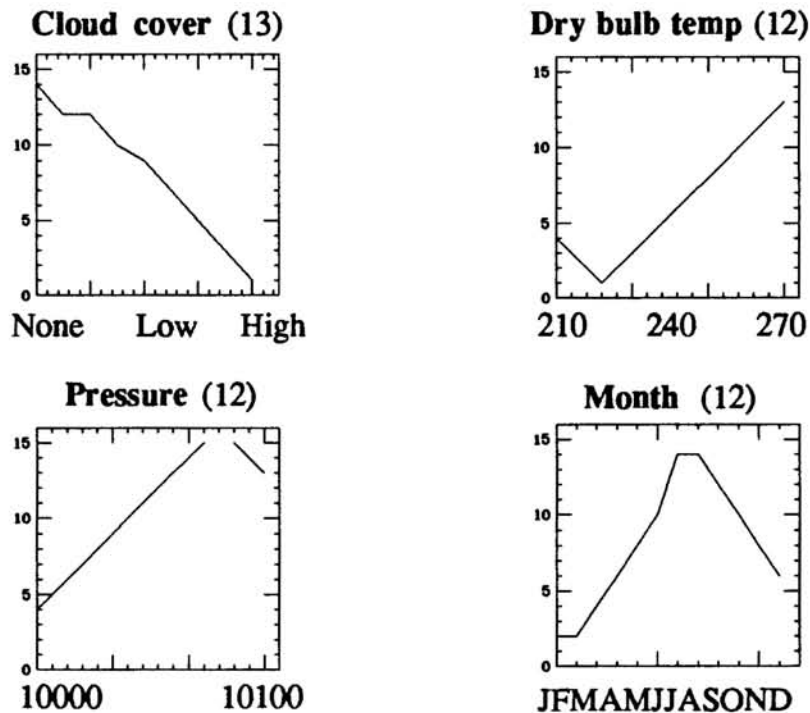

**Figure 6**: The four most sensitive features

The "most preferred values" of these fields are the *minima* of these graphs. For exam-
ple, this location greatly prefers January and February over June and July. The pref-
erences of this location are for the month to be January or February, for low pres-
sure, high cloud cover, and low temperature. Surprisingly, whether it rained in the
*last* four hours is not one of the most important features for this location.

We can also look some of the least sensitive features. The following graphs show
the least sensitive features of the memory location towards predicting rain; that is,
the location is relatively insensitive to the values of these features.

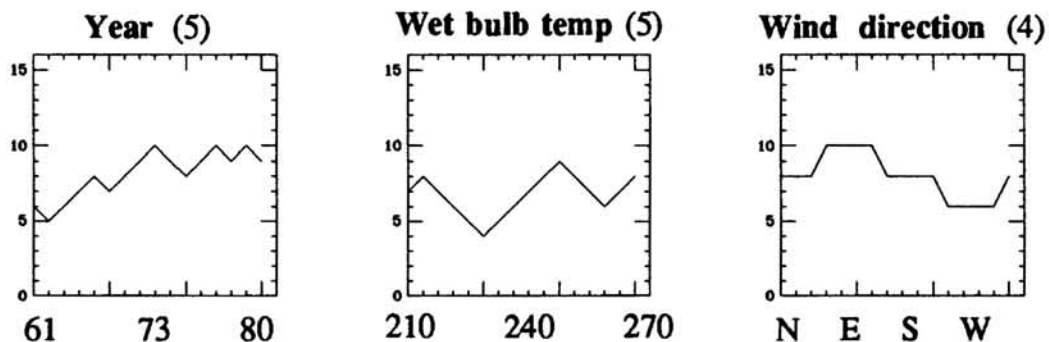

**Figure 7**: The three least sensitive features

This set contains some fields that one *would* expect to be relatively unimportant,
such as year. Fields such as wind direction is unimportant to this location, but inter-
estingly other highly-rated locations find it to be very useful in other regions of the
weather space.

## COMPARISON WITH DAVIS' METHOD

Davis' Algorithm has been shown to be a powerful new method for augmenting the power of a backpropagation-based system. The following is an attempt to contrast our approaches, without denigrating the importance his groundbreaking work. The reader is referred to his book for detailed information about his approach.

It is difficult to directly compare the performance of these techniques given the preliminary nature of the experiments done with Genetic Memory. However, it is possible to compare architectural features of the systems, and estimate the relative strengths a weaknesses.

• **Backpropagation vs. Associative Memories:** Davis' approach relies on the performance of the backpropagation algorithm for the central learning cycle of the system. Associative memories have a far quicker learning cycle than backpropagation networks, and have been shown to have preferential characteristics after training in some domains. A system based on an associative memory may share these advantages over a system based on backpropagation.

• **Scalability:** Many issues concerning the scalability of backpropagation networks remain unresolved. It is not simple to build backpropagation networks of thousands or hundreds of thousands of units. In contrast, Kanerva's Sparse Distributed Memory is specifically designed for such massive construction; one implementation on the Connection Machine can contain 1,000,000 hidden units. The Genetic Memory shares this property.

• **Unity:** Davis' algorithm has two levels of processing. The first level consists of standard backpropagation networks, and the second is a meta-level which manipulates these networks. The Genetic Memory has incorporated both algorithms into a single network; both algorithms are operating simultaneously.

My intuition is that *different algorithms may be best suited for the different layers of a neural network.* Layers with a large fan-out (such as the input layer to the layer of hidden units) may be best driven by an algorithm suited to high-dimensional searching, such as Genetic Algorithms or a Kohonen-style self-organizing system. Layers with a large fan-in (such as the hidden-unit layer to the output layer) may be best driven by a hill-climbing algorithms such a backpropagation.

## CONCLUSIONS

• Real-world problems are often "high-dimensional", that is, are described by large numbers of dependent variables. Algorithms must be specifically designed to function well in such high-dimensional spaces. Genetic Memory is such an algorithm .

• Genetic Memory, while sharing some features with Davis' approach, has fundamental differences that may make it more appropriate to some problems and easier to scale to extremely-large (> 100,000 node) systems.

• The incorporation of the Genetic Algorithm improves the recall performance of a standard associative memory.

• The structure of the Genetic Memory allows the user to access the parameters discovered by the Genetic Algorithm and used to assist in making the associations stored in the memory.

## Acknowledgments

This work was supported in part by Cooperative Agreements NCC 2-408 and NCC 2-387 from the National Aeronautics and Space Administration (NASA) to the Universities Space Research Association (USRA). Funding related to the Connection Machine was jointly provided by NASA and the Defense Advanced Research Projects Agency (DARPA). All agencies involved were very helpful in promoting this work, for which I am grateful.

The entire RIACS staff and the SDM group has been supportive of my work. Bruno Olshausen was a vital sounding-board. Pentti Kanerva trusted my intuitions even when the payoff wasn't yet clear. And finally, thanks to Doug Brockman, who decided to wait for me.

## References

Davis, L., *Genetic algorithms and simulated annealing*, London, England: Pitman Publishing (1987).

Holland, J. H., *Adaptation in natural and artificial systems*, Ann Arbor: University of Michigan Press (1975).

Holland, J. H., "Escaping brittleness: the possibilities of general-purpose learning algorithms applied to parallel rule-based systems," in *Machine learning, an artificial intelligence approach, Volume II*, R. J. Michalski, J. G. Carbonell, and T. M. Mitchell, eds. Los Altos, California: Morgan Kaufmann (1986).

Kanerva, Pentti., "Self-propagating Search: A Unified Theory of Memory," Center for the Study of Language and Information Report No. CSLI-84-7 (1984).

Kanerva, Pentti., *Sparse distributed memory*, Cambridge, Mass: MIT Press, 1988.

Rogers, David, "Using data-tagging to improve the performance of Kanerva's sparse distributed memory," Research Institute for Advanced Computer Science Technical Report 88.1, NASA Ames Research Center (1988a).

Rogers, David, "Kanerva's Sparse Distributed Memory: an Associative Memory Algorithm Well-Suited to the Connection Machine," *Int. J. High-Speed Comput.*, 2, pp. 349-365 (1989).

Rogers, David, "Statistical Prediction with Kanerva's Sparse Distributed Memory," *Advances in Neural Information Processing Systems I*, San Mateo: Morgan-Kaufman (1989).